# Modelling Reciprocating Relationships with Hawkes Processes

**Charles Blundell**
Gatsby Computational Neuroscience Unit
University College London
London, United Kingdom
c.blundell@gatsby.ucl.ac.uk

**Katherine A. Heller**
Duke University
Durham, NC, USA
kheller@stat.duke.edu

**Jeffrey M. Beck**
University of Rochester
Rochester, NY, USA
jbeck@bcs.rochester.edu

## Abstract

We present a Bayesian nonparametric model that discovers implicit social structure from interaction time-series data. Social groups are often formed implicitly, through actions among members of groups. Yet many models of social networks use explicitly declared relationships to infer social structure. We consider a particular class of Hawkes processes, a doubly stochastic point process, that is able to model reciprocity between groups of individuals. We then extend the Infinite Relational Model by using these reciprocating Hawkes processes to parameterise its edges, making events associated with edges co-dependent through time. Our model outperforms general, unstructured Hawkes processes as well as structured Poisson process-based models at predicting verbal and email turn-taking, and military conflicts among nations.

## 1 Introduction

As social animals, people constantly organise themselves into social groups. These social groups can revolve around particular activities, such as sports teams, particular roles, such as store managers, or general social alliances, like gang members. Understanding the dynamics of group interactions is a difficult problem that social scientists strive to address.

One basic problem in understanding group behaviour is that groups are often not explicitly defined, and the members must be inferred. How might we infer these groups, and from what data? How can we predict future interactions among individuals based on these inferred groups?

A common approach is to infer groups, or clusters, of people based upon a declared relationship between pairs of individuals [1, 2, 3, 4]. For example, data from social networks, where two people declare that they are "friends" or in each others' social "neighbourhood", can potentially be used. However these declared relationships are not necessarily readily available, truthful, or pertinent to inferring the social group structure of interest.

In this paper we instead propose an approach to inferring social groups based directly on a set of real interactions between people. This approach reflects an "actions speak louder than words" philosophy. If we are interested in capturing groups that best reflect human behaviour we should be determining the groups from instances of that same behaviour. We develop a model which can learn social group structure based on interactions data.

In the work that we present, our data will consist of a sequence of many events, each event reflecting one person, the sender, performing some sort of an action towards another person, the recipient, at some particular point in time. As examples, the actions we consider are that of one person sending an email to another, one person speaking to another, or one country engaging in military action towards another.

The key property that we leverage to infer social groups is reciprocity. Reciprocity is a common social norm, where one person's actions towards another increases the probability of the same type of action being returned. For example, if Bob emails Alice, it increases the probability that Alice will email Bob in the near future. Reciprocity widely manifests across many cultures, perhaps most commonly as the golden rule and tit for tat retaliation. When multiple people show a similar pattern of reciprocity, our model will place these people in their own group.

The Bayesian nonparametric model we use on these time-series data is generative and accounts for the rate of events between clusters of individuals. It is built upon mutually-exciting point processes, known as Hawkes processes [5, 6]. Pairs of mutually-exciting Hawkes processes are able to capture the causal nature of reciprocal interactions. Here the processes excite one another through their actualised events. Since Poisson processes are a special case of Hawkes processes, our model is also able to capture simpler one-way, non-reciprocal, relationships as well.

Our model is also related to the Infinite Relational Model (IRM) [1, 2]. The IRM typically assumes that there is a fixed graph, or social network, which is observed. Here we are interested in inferring the implicit social structure based only on the occurrences of interactions between vertices in the graph. We apply our model to reciprocal behaviour in verbal and email conversations and to military conflicts among nations.

The remainder of the paper is organised as follows: section 2 discusses using Poisson processes together with the IRM. Section 3 describes our use of self-exciting and pairs of Hawkes processes, and section 4 specifies how they are used to develop our reciprocity clustering model. Section 5 presents an inference algorithm for our model, section 6 discusses related work, and section 7 presents experimental results using our model on synthetic, email, speech and intercountry conflict data.

## 2 Poisson processes with the Infinite Relational Model

The Infinite Relational Model (IRM) [1, 2] was developed to model relationships among entities as graphs, based upon previously declared relationships. Let $V$ denote the vertices of the graph, corresponding to individuals, and let $e_{uv}$ denote the presence or absence of a relationship between vertices $u$ and $v$, corresponding to an edge in the graph. The generative process of the IRM is:

$$\pi \sim \text{CRP}(\alpha) \tag{1}$$
$$\lambda_{pq} \sim \text{Beta}(\gamma, \gamma) \qquad\qquad \forall p, q \in \text{range}(\pi) \tag{2}$$
$$e_{uv} \sim \text{Bernoulli}(\lambda_{\pi(u)\pi(v)}) \qquad\qquad \forall u, v \in V \tag{3}$$

where $\pi$ is a partition of the vertices $V$, distributed according to the Chinese restaurant process (CRP) with concentration parameter $\alpha$, with $p$ and $q$ indexing clusters of $\pi$. Hence vertex $u$ belongs to the cluster given by $\pi(u)$, and consequently, the clusters in $\pi$ are given by $\text{range}(\pi)$. The probability of an edge between vertex $u$ and vertex $v$ is then the parameter $\lambda_{pq}$ associated with their pair of clusters.

Often in interaction data there are many instances of interactions between the same pair of individuals–this cannot be modelled by the IRM. A straightforward way to modify the IRM to account for this is to use a Gamma-Poisson observation model instead of this usual Beta-Bernoulli model. Unfortunately, a vanilla Gamma-Poisson observation model does not allow us to predict events into the future, outside the observed time window. Therefore we consider using a Poisson process instead.

Poisson processes are stochastic counting processes. For an introduction see [7]. We shall consider Poisson processes on $[0, \infty)$, such that the number of events in any interval $[s, s')$ of the real-half line, denoted $N[s, s')$, is Poisson distributed with rate $\lambda(s' - s)$.

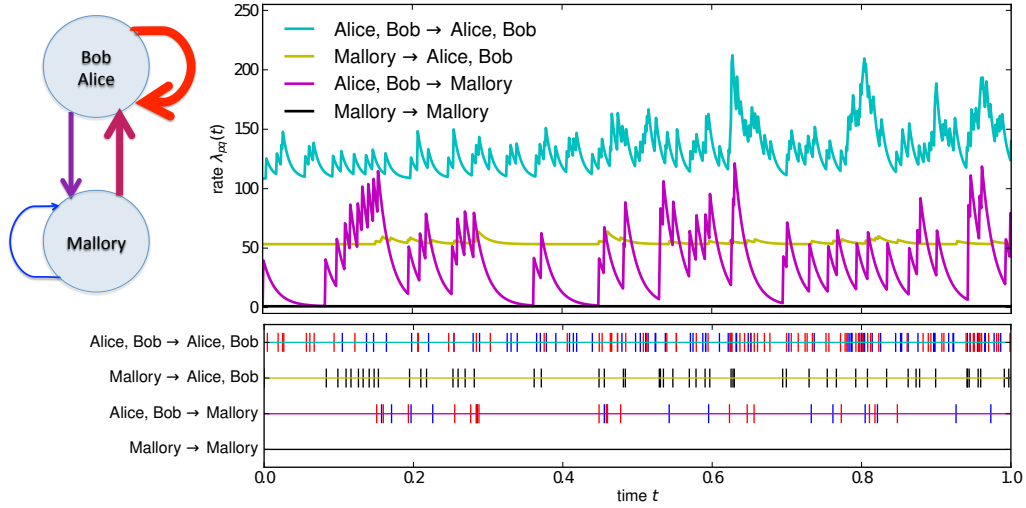

Figure 1: A simple example. The graph in the top left shows the clusters and edge weights learned by our model from the data in the bottom right plot. The top right plot shows the rates of interaction events between clusters. The bottom right plot shows the interaction events. In the graph, the width and temperature (how red the colour is) denotes the expected rate of events between pairs of clusters (using equations (9) and (10)). While in plots on the right, line colours indicates the identity of cluster pairs, and box colours indicate the originator of the event: Alice (red), Bob (blue), Mallory (black). Alice and Bob interact with each other such that they positively reciprocate each others' actions. Mallory, however, has an asymmetric relationship with both Alice and Bob. Only after many events caused by Mallory do Alice or Bob respond, and when they do respond they both, similarly, respond more sparsely.

With Gamma priors on the rate parameter, the full Poisson Process IRM model is:

$$\pi \sim \text{CRP}(\alpha) \tag{4}$$

$$\lambda_{pq} \sim \text{Gamma}(\delta, \beta) \qquad \forall p, q \in \text{range}(\pi) \tag{5}$$

$$N_{uv}(\cdot) \sim \text{PoissonProcess}(\lambda_{\pi(u)\pi(v)}) \qquad \forall u, v \in V \tag{6}$$

where $N_{uv}(\cdot)$ is the random counting measure of the Poisson process, and $\delta$ and $\beta$ are respectively the shape and inverse scale parameters of the Gamma prior on the rate of the Poisson processes, $\lambda_{pq}$.

Inference proceeds by conditioning on, $N_{uv}[0, T] = n_{uv}$ where $n_{uv}$ is the total number of events directed from $u$ to $v$ in the given time interval. Since conjugacy can be maintained, due to the superposition property of Poisson processes, inference in this model is possible in much the same way as in the original IRM [2, 1].

There are two notable deficiencies of this model: the rate of events on each edge is independent of every other edge, and conditioned on the time interval containing all observed events, the times of these events are uniformly distributed. This is not the typical pattern we observe in interaction data. If I send an email to someone, it is more likely that I will receive an email from them than had I not sent an email, and the probability of receiving a reply decreases as time advances. In the following sections we will introduce and utilise mutually-exciting Hawkes processes, which are able to exactly model these phenomena.

## 3 Self-Exciting and Pairs of Mutually-Exciting Hawkes Processes

Hawkes [5, 6] introduced a family of self- and mutually-exciting Markov point processes, often called Hawkes processes. These processes are intuitively similar to Poisson processes, but unlike Poisson processes, the rates of Hawkes processes depend upon their own historic events and those of other processes in an excitatory fashion.

We shall consider an array of $K \times K$ Hawkes processes, where $K$ is the number of clusters in a partition drawn from a CRP restricted to the individuals $V$. As in the IRM, the CRP allows the

number of processes to grow in an unconstrained manner as the number of individuals in the graph grows. However, unlike the IRM, these Hawkes processes will be pairwise-dependent: the Hawkes process governing events from cluster $p$ to cluster $q$, will depend upon the Hawkes process governing events from cluster $q$ to cluster $p$.

Let $N_{pq}$ be the counting measure of the $(p, q)$th Hawkes process. Each Hawkes process is a point process whose rate at time $t$ is given by:

$$\lambda_{pq}(t) = \gamma_{pq} n_p n_q + \int_{-\infty}^{t} g_{pq}(t-s) \mathrm{d}N_{qp}(s) \tag{7}$$

where $\gamma_{pq}$ is the base rate of the counting measure of the Hawkes, process, $N_{pq}$. $n_p$ and $n_q$ are the number of individuals in cluster $p$ and $q$ respectively, and $g_{pq}$ is a non-negative function such that $\int_0^{\infty} g_{pq}(s)ds < 1$, ensuring that $N_{pq}$ is stationary. $N_{qp}$ is the counting measure of the reciprocating Hawkes process of $N_{pq}$. Intuitively, if $N_{pq}$ governs events from cluster $p$ to cluster $q$, then $N_{qp}$ governs events from cluster $q$ to cluster $p$. Equation (7) shows how the rates of events in these two processes are intimately intertwined.

Since $N_{qp}$ is an atomic measure, whose atoms correspond to the times of events, we can express the rate of $N_{pq}$ given in (7), by conditioning on the events of its reciprocating processes $N_{qp}$, as:

$$\lambda_{pq}(t) = \gamma_{pq} n_p n_q + \sum_{i:t_i^{qp} < t} g_{pq}(t - t_i^{qp}) \tag{8}$$

where $t_i^{qp}$ denotes the times of the $i$th event of process $N_{qp}$. Thus the rate of the process $N_{pq}$ at time $t$ is some base rate at which events occur, $\gamma_{pq}$, plus an additional rate of $g_{pq}(t - t_i^{qp})$ for each event in the reciprocating process $N_{qp}$. Figure 1(top) shows an example of how $\lambda_{pq}(t)$ and $\lambda_{qp}(t)$ vary for these pairs of processes.

If $g_{pq}(\cdot) = 0$ then the process is a Poisson process with rate $\gamma_{pq} n_p n_q$. When $p = q$, the process is self-exciting: its current rate depends solely on its own previous events. In our application, self-exciting processes model interactions within a social group, as they model cohesion in reciprocity: individual reciprocation within a group is as if towards oneself. In the case of $p \neq q$, each pair of processes $N_{pq}$ and $N_{qp}$ mutually excite one another. An event in one increases the probability of an event from the other, and so on. Importantly, the type of reciprocation (parameterised by $g_{pq}$ and $g_{qp}$, respectively) differs between events from group $p$ to group $q$ and events from group $q$ to group $p$. This difference in reciprocity is what we would like our model to leverage to learn about social groups.

Hawkes processes are an example of doubly stochastic point processes. The rate of events is itself a random variable. By integrating out the events of $N_{qp}$ we can see that this process is stationary, as its rate does not depend upon time, and also gain further insight into the role of the functions $g_{pq}$ and $g_{qp}$. For self-exciting Hawkes processes, where $p = q$, the marginal rate is:

$$\mathbb{E}[\lambda_{pp}(t)] = n_p^2 \frac{\gamma_{pp}}{1 - G_{pp}} \tag{9}$$

whilst for a pair of mutually-exciting Hawkes processes the marginal rate is:

$$\mathbb{E}[\lambda_{pq}(t)] = n_p n_q \frac{\gamma_{pq} + \gamma_{qp} G_{pq}}{1 - G_{pq} G_{qp}} \tag{10}$$

where $G_{pq} = \int_{-\infty}^{t} g_{pq}(t-u)du$ which tempers the effect of the rate of events from one process on the rate of the other. The closer $G_{pq}$ is to zero, the more Poisson-like Hawkes processes behave. Whilst as $G_{pq}$ approaches one, the rate of events in $N_{pq}$ are increasingly caused by those in $N_{qp}$.

## 4 Hawkes Processes with the Infinite Relational Model

We combine Hawkes processes with the IRM as follows. We pick the form for the $g_{pq}$ functions as $g_{pq}(\delta) = \beta_{pq} e^{-\frac{\delta}{\tau_{pq}}}$ [5, 6, 8, 9]. Examples of using this parameterisation are shown in Figure 1(top).

Due to the memorylessness property of the exponential distribution, inference with Hawkes processes with this parameterisation takes time linear in the number of events [10]. Our generative model is as follows:

$$\pi \sim \text{CRP}(\alpha) \tag{11}$$

$$\lambda_{pq}(t) = \gamma_{pq} n_p n_q + \beta_{pq} \int_{-\infty}^{t} e^{-\frac{t-s}{\tau_{pq}}} \mathrm{d}N_{qp}(s) \qquad \forall p, q \in \text{range}(\pi) \tag{12}$$

$$N_{pq}(\cdot) \sim \text{HawkesProcess}(\lambda_{pq}(\cdot)) \tag{13}$$

$$N_{uv}(\cdot) \sim \text{Thinning}(N_{\pi(u)\pi(v)}(\cdot)) \qquad \forall u, v \in V \tag{14}$$

where, as before, $\pi$ is a partition of the individuals, drawn from a Chinese restaurant process (CRP) with concentration parameter $\alpha$. For each pair of clusters $p$ and $q$, we associate a time-varying rate $\lambda_{pq}(t)$ which dictates the rate of events from individuals in cluster $p$ to individuals in cluster $q$, and a Hawkes process $N_{pq}$. As described in the previous section, this rate depends upon the specific events sent in the opposite direction, from cluster $q$ to cluster $p$, whose measure is also random and is denoted $N_{qp}(\cdot)$.

Each random measure $N_{uv}(\cdot)$ governs events between a particular pair of individuals within clusters $p$ and $q$ respectively. $N_{uv}(\cdot)$ are drawn by thinning the cluster random measure $N_{pq}(\cdot)$ among all of the edges between individuals in clusters $p$ and $q$. Thinning means distributing the atoms of $N_{pq}(\cdot)$ among each $N_{uv}(\cdot)$, such that $N_{pq} = \sum_{u,v} N_{uv}(\cdot)$. Constructing the edge measures by thinning means it is sufficient to ensure that $\int_0^\infty g_{pq}(u)du < 1$ for the process to be stationary. This condition, under the chosen parameterisation, implies that $\tau_{pq}\beta_{pq} < 1$. When all $\beta_{pq} = 0$, this model is equivalent to the Poisson process IRM in section 2. Henceforth we will use uniform thinning—each event in $N_{pq}(\cdot)$ is assigned uniformly at random among all $N_{uv}(\cdot)$ where $p = \pi(u)$ and $q = \pi(v)$—but in principle any thinning scheme may be used.

For a Hawkes process $N_{pq}$, the rate at which no events occurs in the interval $[s, s')$ is:

$$e^{-\int_s^{s'} \lambda_{pq}(t)\mathrm{d}t} \tag{15}$$

Suppose we observe the times of all the events in $[0, T)$, $\{t_i^{uv}\}_{i=1}^{n_{uv}}$ for process $N_{uv}$ ($n_{uv}$ being the total number of events from $u$ to $v$ in $[0, T)$). Suppose that individual $u$ is in cluster $p$ and that individual $v$ is in cluster $q$. Furthermore, assume there are no events before time 0. The likelihood of each edge between individuals $u$ and $v$ is thus:

$$p(\{t_i^{uv}\}_{i=1}^{n_{uv}}|\theta_{pq}, \{t_i^{qp}\}_{i=1}^{n_{qp}}) = e^{-\frac{1}{n_p n_q} \int_0^T \lambda_{pq}(t)\mathrm{d}t} \prod_{i=1}^{n_{uv}} \frac{\lambda_{pq}(t_i^{uv})}{n_p n_q} \tag{16}$$

where $\theta_{pq} = (\gamma_{pq}, \beta_{pq}, \tau_{pq})$, $\{t_i^{qp}\}_{i=1}^{n_{qp}}$ are the times of the reciprocal events. We place proper uniform priors on $\log \alpha$, $\gamma_{pq}$, $\beta_{pq}$, and $\tau_{pq}$, enforcing the constraint that $\tau_{pq}\beta_{pq} < 1$.

## 5 Inference

We perform posterior inference using Markov chain Monte Carlo. Our model is a departure from previous IRM-based models as there is no conjugate prior for the likelihood. Thus we cannot simply integrate out these parameters, and must sample them.

To infer the partition of individuals $\pi$, the concentration parameter $\alpha$, and the parameters of each Hawkes process $\theta_{pq} = (\gamma_{pq}, \beta_{pq}, \tau_{pq})$, we use Algorithm 5 [11] adapted to the IRM and slice sampling [12] to draw samples from the posterior. We initialise the chain from the prior. Slice sampling is used for $\alpha$ and each of $\gamma_{pq}$, $\beta_{pq}$, and $\tau_{pq}$. When setting the bounds of the slice sampler for $\beta_{pq}$ ($\tau_{pq}$) we set the upper bound to $\frac{1}{\tau_{pq}}$ ($\frac{1}{\beta_{pq}}$) respectively, to ensure that $\beta_{pq}\tau_{pq} \leq 1$.

## 6 Related work

Several authors have considered modelling occurrence events [13, 14, 15] using piecewise constant rate Markov point processes for known number of event types. Our work directly models interaction

events (where an event is structured to have a sender and recipient) and the number of possible events types is not limited. [16] describes a model of occurrence events as a discrete time-series using a latent first-order Markov model. Our model differs in that it considers interaction events in continuous time and requires no first-order assumption.

The model in Section 2 relates the work of [17] to the IRM [1], yielding a version of their model that learns the number of clusters whilst maintaining conjugacy. However our model does not use a Poisson process to model event times, instead using processes which have a time-varying rate.

Simma and Jordan [10] describe a cascade of Poisson processes, forming a marked Hawkes process. Hawkes processes are also the basis of this work, however our work does not use side-channel information to group individuals by imposing fixed marks on the process; instead we learn structure among several co-dependent Hawkes processes and use Bayesian inference for the parameters and structure.

Paninski et al [18, 19] describe a process similar to a Hawkes process that uses an additional link function to allow for inhibition amongst neurons. The interest is in modelling the activation and co-activation of neurons and as such they do not directly model cluster structure among the neurons, while our model does model this structure. Learning such structure among neurons is a potential interesting future application of this model.

Our model may also be seen as a probabilistic interpretation of the interaction rank of [20], which we leverage to discover global clustering structure. An interesting future direction would be to learn a per-person (i.e., ego-centric) clustering structure.

## 7   Experiments

In our experiments we compared our model to the Poisson process IRM (Section 2), a single Hawkes process and a single Poisson process. These latter two models are equivalent to the first two models where just one cluster is used.

We compared these models quantitatively by comparing their log predictive densities (with respect to the space of ordered sequences of events) on events falling in the final 10% of the total time of the data (Table 2). We normalised the times of all events such that the first 90% of total time lay in the interval $[0, 1]$. We ran our inference algorithm for 5000 iterations and discarded the first 500 burn-in samples, repeating each experiment 10 times from different initialisations from the prior.

**Synthetic data**   We generated synthetic data to highlight differences between our model and the alternatives. The data involves three individuals and is plotted in Figure 1. Table 1 shows details of the fit of the model to the data, and Table 2 shows the predictive results. The Poisson IRM is uncertain how to cluster individuals as it cannot model the temporal dependence between individuals, while the Hawkes IRM can and so performs better at prediction as well. A single Hawkes process does not model the structure among individuals and so performs worse than the Hawkes IRM, although it is able to model dependence among events.

**Enron email threads**   We took the five longest threads from the Enron 2009 data set [21]. We identified threads by the set of senders and receivers, and their subject line (after removing common subject line prefixes such as "Re:", "Fwd:" and so on, removing punctuation and making all letters lower case). All of these threads involve two different people so there is little scope for learning much group structure in these data: either both people are in the same cluster, or they are in two separate clusters. However as can be seen in Table 2 these data suggest a predictive advantage to using mutually-exciting Hawkes processes, as automatically determined by our model, instead of a single self-exciting Hawkes process and of both of these approaches over their corresponding Poisson processes model. A self-exciting Hawkes process is unable to mark the sender and receiver of events as differing, whilst Poisson process-based models are unable to model the causal structure of events.

**Santa Barbara Conversation Corpus**   We took five conversations from the Santa Barbara Conversation Corpus [22] involving the largest number of people. These results are labelled "SB conv" followed by the conversation identifier in this corpus, in the results in Tables 1 and 2. These con-

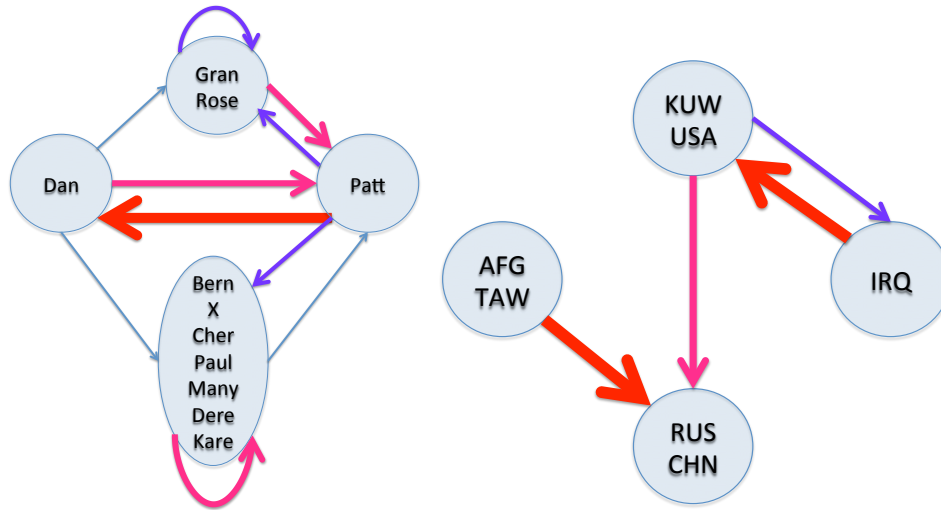

Figure 2: Graphs of clusters of individuals inferred by our model. Edge width and temperature (how red the colour is) denotes the expected rate of events between pairs of clusters (using equations (9) and (10); edges whose marginal rate is below 1 are not included). On the left is the graph inferred on the "SB conv 26" data set. On the right is the graph inferred on the "Small MID" data set.

versations cover a variety of social situations: questions during a university lecture (12), a book discussion group (23), a meeting among city officials (26), a family argument/discussion (33), and a conversation at a family birthday party (49). We modelled the turn-taking behaviour of these groups by taking the times of when one speaker switched to the next. In Figure 2(left) we show the cluster graph found by our model for conversation 26, involving city officials discussing a grant application. The identities of participants in all of these data are anonymised, preventing an exact interpretation. However, the model captures the discussive to-and-fro of the meeting, where PATT appears to be the chair of the meeting, and DAN, ROSE and GRAN are the main discussors, all of whom discuss with the chair, initially in question and an answer format, and among themselves, with other members of the audience chipping in sporadically.

**Correlates of war**  We use version 3.0 of the Militarized Interstate Disputes (MIDs) data set [23] to model correlates of war. This data set spans the years 1993 to 2001, and consists of MID incidents, along with the countries involved in the incidents. Incidents vary from diplomatic threats of military force to the actual deployment of military force against another state. A detailed description of each incident is available in [24].

The results of all models on the correlates of war data are given in Table 2 with details of the fits in Table 1 in the rows entitles "Small MID" and "Full MID". The full MID data set consists of 82 countries—yielding a large graph. For exposition purposes, we show the graph (in Figure 2(right)) on part of the MID data set, by restricting to events among the USA, Kuwait, Afghanistan, Taiwan, Russia, China, and Iraq. Thicker and redder lines between clusters (computed from equations 9 and 10) reflect a higher rate of incidents directed between the countries along the edge.

The results of the clustering given by our model are in keeping with that discussed in [24]. There were three main conflicts involving the countries we modelled during the time period this data covers. These conflicts involve 1) Russia and Afghanistan, 2) Taiwan (sometimes with support from the USA) and China (sometimes with support from Russia), and 3) Iraq, Kuwait, and the USA. 1) Revolved mostly around border disputes coming out of the Soviet war in Afghanistan, and incidents sometimes involved using former Soviet countries as proxies. 2) Reflects conflict between Taiwan and China over potential Taiwanese independence. Lastly, 3) deals with conflicts between Iraq and either Kuwait or the USA coming out of the Persian Gulf war. It is interesting to note that groups involving smaller countries were found to be more likely to initiate incidents with larger countries in a dispute (e.g. Iraq was almost always the instigator of disputes in their conflict with Kuwait and the USA). Since the data ends in 2001, relatively few disputes with Afghanistan involve the USA.

| | N | T | Hawkes IRM $\mathbb{E}[K]$ | Hawkes IRM log probability | Poisson IRM $\mathbb{E}[K]$ | Poisson IRM log probability |
|---|---|---|---|---|---|---|
| Synthetic | 3 | 239 | 2.00 | 594.04±0.01 | 1.36 | 533.65±0.00 |
| Small MID | 7 | 57 | 4.30 | 33.59±0.02 | 1.02 | -63.99±0.03 |
| Full MID | 82 | 412 | 13.67 | -638.25±1.16 | 3.93 | -1412.49±5.38 |
| Enron 0 | 2 | 896 | 2.00 | 6724.76±0.01 | 2.00 | 4516.77±0.00 |
| Enron 1 | 2 | 204 | 2.00 | 1202.99±0.02 | 2.00 | 692.32±0.00 |
| Enron 2 | 2 | 122 | 2.00 | 616.37±0.02 | 2.00 | 336.02±0.00 |
| Enron 3 | 2 | 117 | 2.00 | 497.53±0.02 | 2.00 | 318.38±0.00 |
| Enron 4 | 2 | 85 | 2.00 | 252.60±0.02 | 2.00 | 192.74±0.00 |
| SB conv 23 | 18 | 832 | 11.87 | 1581.72±0.12 | 3.01 | 599.29±0.42 |
| SB conv 26 | 11 | 95 | 4.26 | 170.34±0.03 | 2.00 | -51.92±0.14 |
| SB conv 12 | 12 | 133 | 4.11 | 233.41±0.03 | 2.53 | -59.12±0.15 |
| SB conv 49 | 11 | 620 | 8.85 | 1728.13±0.07 | 3.40 | 990.75±0.15 |
| SB conv 33 | 10 | 499 | 8.44 | 803.22±0.16 | 2.03 | 431.59±0.12 |

Table 1: Details of data sets and fits of the structured models. $N$ denotes the number of individuals in the data set. $T$ denotes the total number of events in the data set. $\mathbb{E}[K]$ is the average number of clusters found in the posterior. Log probability is the average log probability of the training data.

| | Hawkes IRM | Poisson IRM | Hawkes | Poisson |
|---|---|---|---|---|
| Synthetic | **43.00±0.00** | -6.76±0.02 | 39.88±0.01 | -3.88±0.00 |
| Small MID | **12.69±0.04** | -50.88±0.02 | 6.37±0.01 | -50.86±0.00 |
| Full MID | **-134.97±2.98** | -355.29±5.61 | -188.08±0.00 | -302.65±0.00 |
| Enron 0 | **259.20±0.01** | 39.33±0.00 | 233.44±0.00 | 40.11±0.00 |
| Enron 1 | **436.66±0.01** | 133.29±0.00 | 380.27±0.01 | 105.71±0.00 |
| Enron 2 | **139.40±0.01** | 24.14±0.00 | 118.86±0.00 | 22.88±0.00 |
| Enron 3 | **124.22±0.01** | 21.06±0.00 | 101.71±0.01 | 21.03±0.00 |
| Enron 4 | **127.82±0.02** | 28.38±0.00 | 109.62±0.00 | 22.08±0.00 |
| SB conv 23 | **132.57±0.27** | -198.34±0.23 | 30.93±0.00 | -213.18±0.00 |
| SB conv 26 | **-5.85±0.02** | -16.83±0.09 | -6.05±0.00 | -14.54±0.00 |
| SB conv 12 | **96.07±0.03** | -97.89±0.10 | 33.18±0.00 | -128.53±0.00 |
| SB conv 49 | **220.85±0.09** | -116.62±0.12 | 126.94±0.00 | -83.62±0.00 |
| SB conv 33 | **46.19±0.06** | -100.83±0.04 | 21.71±0.00 | -83.79±0.00 |

Table 2: Average log predictive results for each model with standard errors

# 8 Discussion

We have presented a Bayesian nonparametric approach to learning the structure among collections of co-dependent Hawkes processes, which on several interaction data sets consistently outperforms both unstructured and Poisson-based models in terms of predictive likelihoods. The intuition behind why our model works well is that it captures part of the reciprocal nature of interactions among individuals in social situations, which in turn requires modelling some of the causal relationship of events. By learning this structure, our model is able to make better predictions.

There are several future directions. For example, individuals might contribute to groups differently to one another. There may be different kinds of events between individuals and other side-channel information. Both of these artefacts may be modelled by replacing the uniform thinning scheme proposed above, with a detailed model of these effects. It would be interesting to consider other parameterisations of $g_{pq}(\cdot)$ that, for example, include periods of delay between reciprocation; the exponential parameterisation lends itself to efficient computation [10] whilst other parameterisations do not necessarily have this property. But different choices of $g_{pq}(\cdot)$ may yield better statistical models. Another interesting avenue is to explore other structure amongst interaction events using Hawkes processes, beyond reciprocity.

**Acknowledgements** The authors are grateful for helpful comments from the anonymous reviewers, and the support of Josh Tenenbaum, the Gatsby Charitable Foundation, PASCAL2 NoE, NIH award P30 DA028803, and an NSF postdoctoral fellowship,

# References

[1] Charles Kemp, Joshua B. Tenenbaum, Thomas L. Griffiths, Takeshi Yamada, and Naonori Ueda. Learning systems of concepts with an infinite relational model. *AAAI*, 2006.

[2] Zhao Xu, Volker Tresp, Kai Yu, and Hans-Peter Kriegel. Infinite hidden relational models. *Uncertainty in Artificial Intelligence (UAI)*, 2006.

[3] Edoardo M. Airoldi, David M. Blei, Stephen E. Fienberg, and Eric P. Xing. Mixed membership stochastic blockmodel. *Journal of Machine Learning Research*, 9:1981–2014, 2008.

[4] Konstantina Palla, David A. Knowles, and Zoubin Ghahramani. An infinite latent attribute model for network data. In *Proceedings of the 29th International Conference on Machine Learning*, ICML 2012. July 2012.

[5] Alan G. Hawkes. Point spectra of some self-exciting and mutually-exciting point processes. *Journal of the Royal Statistical Society. Series B (Methodological)*, 58:83–90, 1971.

[6] Alan G. Hawkes. Point spectra of some mutually-exciting point processes. *Journal of the Royal Statistical Society. Series B (Methodological)*, 33(3):438–443, 1971.

[7] John F. C. Kingman. *Poisson Processes*. Oxford University Press, 1993.

[8] Alan G. Hawkes and David Oakes. A cluster process representation of a self-exciting process. *Journal of Applied Probability*, 11(3):493–503, 1974.

[9] David Oakes. The Markovian self-exciting process. *Journal of Applied Probability*, 12(1):69–77, 1975.

[10] Aleskandr Simma and Michael I. Jordan. Modeling events with cascades of poisson processes. *Uncertainty in Artificial Intelligence (UAI)*, 2010.

[11] Radford M. Neal. Markov chain sampling methods for Dirichlet process mixture models. Technical Report 9815, University of Toronto, 1998.

[12] Radford M. Neal. Slice sampling. *Annals of Statistics*, 31(3):705767, 2003.

[13] Uri Nodelman, Christian R. Shelton, and Daphne Koller. Continuous time Bayesian networks. *Uncertainty in Artificial Intelligence (UAI)*, 2002.

[14] Shyamsundar Rajaram, Thore Graepel, and Ralf Herbrich. Poisson-networks: A model of structured point processes. *Proceedings of the Tenth International Workshop on Artificial Intelligence and Statistics (AISTATS)*, 2005.

[15] Asela Gunawardana, Christopher Meek, and Puyang Xu. A model for temporal dependencies in event streams. *Neural Information Processing Systems (NIPS)*, 2011.

[16] David Wingate, Noah D. Goodman, Daniel M. Roy, and Joshua B. Tenenbaum. The infinite latent events model. *Uncertainty in Artificial Intelligence (UAI)*, 2009.

[17] Christopher DuBois and Padhraic Smyth. Modeling relational events via latent classes. In *Proceedings of the 16th ACM SIGKDD Conference on Knowledge Discovery and Data Mining*, 2010.

[18] Liam Paninski. Maximum likelihood estimation of cascade point-process neural encoding models. *Network*, 2004.

[19] Liam Paninski, Jonathan Pillow, and Jeremy Lewi. Statistical models for neural encoding, decoding, and optimal stimulus design. In *Computational Neuroscience: Theoretical Insights Into Brain Function*. 2007.

[20] Maayan Roth, Assaf Ben-David, David Deutscher, Guy Flysher, Ilan Horn, Ari Leichtberg, Naty Leiser, Yossi Matias, and Ron Merom. Suggesting friends using the implicit social graph. In *Proceedings of the 16th ACM SIGKDD Conference on Knowledge Discovery and Data Mining*, 2010.

[21] Enron 2009 Data set. http://www.cs.cmu.edu/ enron/.

[22] John W. DuBois, Wallace L. Chafe, Charles Meyer, and Sandra A. Thompson. *Santa Barbara corpus of spoken American English*. Linguistic Data Consortium, 2000.

[23] Faten Ghosn, Glenn Palmer, and Stuart Bremer. The mid3 data set, 19932001: Procedures, coding rules, and description. *Conflict Management and Peace Science*, 21:133–154, 2004.

[24] Dispute Narratives. http://www.correlatesofwar.org/cow2%20data/mids/mid_v3.0.narratives.pdf.

